# The Rectified Gaussian Distribution

**N. D. Socci, D. D. Lee and H. S. Seung**
Bell Laboratories, Lucent Technologies
Murray Hill, NJ 07974
{nds|ddlee|seung}@bell-labs.com

## Abstract

A simple but powerful modification of the standard Gaussian distribution is studied. The variables of the rectified Gaussian are constrained to be nonnegative, enabling the use of nonconvex energy functions. Two multimodal examples, the competitive and cooperative distributions, illustrate the representational power of the rectified Gaussian. Since the cooperative distribution can represent the translations of a pattern, it demonstrates the potential of the rectified Gaussian for modeling pattern manifolds.

## 1  INTRODUCTION

The *rectified* Gaussian distribution is a modification of the standard Gaussian in which the variables are constrained to be nonnegative. This simple modification brings increased representational power, as illustrated by two multimodal examples of the rectified Gaussian, the competitive and the cooperative distributions. The modes of the competitive distribution are well-separated by regions of low probability. The modes of the cooperative distribution are closely spaced along a nonlinear continuous manifold. Neither distribution can be accurately approximated by a single standard Gaussian. In short, the rectified Gaussian is able to represent both discrete and continuous variability in a way that a standard Gaussian cannot.

This increased representational power comes at the price of increased complexity. While finding the mode of a standard Gaussian involves solution of linear equations, finding the modes of a rectified Gaussian is a quadratic programming problem. Sampling from a standard Gaussian can be done by generating one dimensional normal deviates, followed by a linear transformation. Sampling from a rectified Gaussian requires Monte Carlo methods. Mode-finding and sampling algorithms are basic tools that are important in probabilistic modeling.

Like the Boltzmann machine[1], the rectified Gaussian is an undirected graphical model. The rectified Gaussian is a better representation for probabilistic modeling

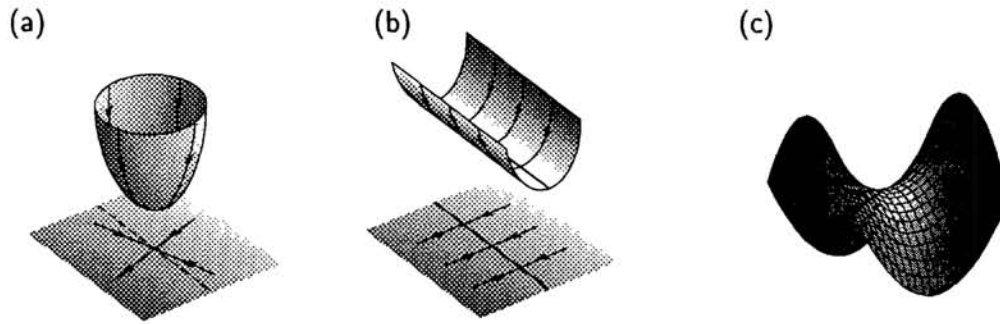

Figure 1: Three types of quadratic energy functions. (a) Bowl (b) Trough (c) Saddle

of continuous-valued data. It is unclear whether learning will be more tractable for the rectified Gaussian than it is for the Boltzmann machine.

A different version of the rectified Gaussian was recently introduced by Hinton and Ghahramani[2, 3]. Their version is for a single variable, and has a singularity at the origin designed to produce sparse activity in directed graphical models. Our version lacks this singularity, and is only interesting in the case of more than one variable, for it relies on undirected interactions between variables to produce the multimodal behavior that is of interest here.

The present work is inspired by biological neural network models that use continuous dynamical attractors[4]. In particular, the energy function of the cooperative distribution was previously studied in models of the visual cortex[5], motor cortex[6], and head direction system[7].

## 2   ENERGY FUNCTIONS: BOWL, TROUGH, AND SADDLE

The standard Gaussian distribution $P(x)$ is defined as

$$P(x) = Z^{-1}e^{-\beta E(x)} , \tag{1}$$

$$E(x) = \frac{1}{2}x^T A x - b^T x . \tag{2}$$

The symmetric matrix $A$ and vector $b$ define the quadratic energy function $E(x)$. The parameter $\beta = 1/T$ is an inverse temperature. Lowering the temperature concentrates the distribution at the minimum of the energy function. The prefactor $Z$ normalizes the integral of $P(x)$ to unity.

Depending on the matrix $A$, the quadratic energy function $E(x)$ can have different types of curvature. The energy function shown in Figure 1(a) is convex. The minimum of the energy corresponds to the peak of the distribution. Such a distribution is often used in pattern recognition applications, when patterns are well-modeled as a single prototype corrupted by random noise.

The energy function shown in Figure 1(b) is flattened in one direction. Patterns generated by such a distribution come with roughly equal likelihood from anywhere along the trough. So the direction of the trough corresponds to the invariances of the pattern. Principal component analysis can be thought of as a procedure for learning distributions of this form.

The energy function shown in Figure 1(c) is saddle-shaped. It cannot be used in a Gaussian distribution, because the energy decreases without limit down the

sides of the saddle, leading to a non-normalizable distribution. However, certain saddle-shaped energy functions can be used in the rectified Gaussian distribution, which is defined over vectors $x$ whose components are all nonnegative. The class of energy functions that can be used are those where the matrix $A$ has the property $x^T A x > 0$ for all $x > 0$, a condition known as *copositivity*. Note that this set of matrices is larger than the set of positive definite matrices that can be used with a standard Gaussian. The nonnegativity constraints block the directions in which the energy diverges to negative infinity. Some concrete examples will be discussed shortly. The energy functions for these examples will have multiple minima, and the corresponding distribution will be multimodal, which is not possible with a standard Gaussian.

## 3  MODE-FINDING

Before defining some example distributions, we must introduce some tools for analyzing them. The modes of a rectified Gaussian are the minima of the energy function (2), subject to nonnegativity constraints. At low temperatures, the modes of the distribution characterize much of its behavior.

Finding the modes of a rectified Gaussian is a problem in quadratic programming. Algorithms for quadratic programming are particularly simple for the case of non-negativity constraints. Perhaps the simplest algorithm is the projected gradient method, a discrete time dynamics consisting of a gradient step followed by a rectification

$$x_{t+1} = [x_t + \eta(b - Ax_t)]^+ \tag{3}$$

The rectification $[x]^+ = \max(x, 0)$ keeps $x$ within the nonnegative orthant ($x \geq 0$). If the step size $\eta$ is chosen correctly, this algorithm can provably be shown to converge to a stationary point of the energy function[8]. In practice, this stationary point is generally a local minimum.

Neural networks can also solve quadratic programming problems. We define the synaptic weight matrix $W = I - A$, and a continuous time dynamics

$$\dot{x} + x = [b + Wx]^+ \tag{4}$$

For any initial condition in the nonnegative orthant, the dynamics remains in the nonnegative orthant, and the quadratic function (2) is a Lyapunov function of the dynamics.

Both of these methods converge to a stationary point of the energy. The gradient of the energy is given by $g = Ax - b$. According to the Kühn-Tucker conditions, a stationary point must satisfy the conditions that for all $i$, either $g_i = 0$ and $x_i > 0$, or $g_i > 0$ and $x_i = 0$. The intuitive explanation is that in the interior of the constraint region, the gradient must vanish, while at the boundary, the gradient must point toward the interior. For a stationary point to be a local minimum, the Kühn-Tucker conditions must be augmented by the condition that the Hessian of the nonzero variables be positive definite.

Both methods are guaranteed to find a global minimum only in the case where $A$ is positive definite, so that the energy function (2) is convex. This is because a convex energy function has a unique minimum. Convex quadratic programming is solvable in polynomial time. In contrast, for a nonconvex energy function (indefinite $A$), it is not generally possible to find the global minimum in polynomial time, because of the possible presence of local minima. In many practical situations, however, it is not too difficult to find a reasonable solution.

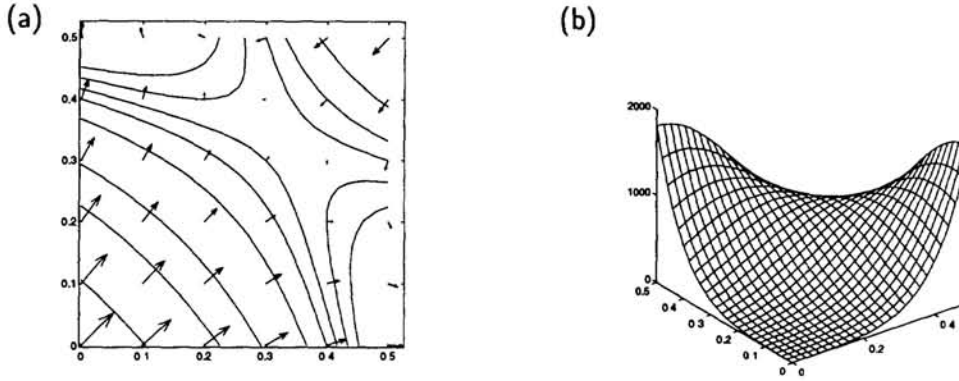

Figure 2: The competitive distribution for two variables. (a) A non-convex energy function with two constrained minima on the $x$ and $y$ axes. Shown are contours of constant energy, and arrows that represent the negative gradient of the energy. (b) The rectified Gaussian distribution has two peaks.

The rectified Gaussian happens to be most interesting in the nonconvex case, precisely because of the possibility of multiple minima. The consequence of multiple minima is a multimodal distribution, which cannot be well-approximated by a standard Gaussian. We now consider two examples of a multimodal rectified Gaussian.

## 4   COMPETITIVE DISTRIBUTION

The competitive distribution is defined by

$$A_{ij} = -\delta_{ij} + 2 \tag{5}$$
$$b_i = 1; \tag{6}$$

We first consider the simple case $N = 2$. Then the energy function given by

$$E(x, y) = -\frac{x^2 + y^2}{2} + (x + y)^2 - (x + y) \tag{7}$$

has two constrained minima at $(1, 0)$ and $(0, 1)$ and is shown in figure 2(a). It does not lead to a normalizable distribution unless the nonnegativity constraints are imposed. The two constrained minima of this nonconvex energy function correspond to two peaks in the distribution (fig 2(b)). While such a bimodal distribution could be approximated by a mixture of two standard Gaussians, a single Gaussian distribution cannot approximate such a distribution. In particular, the reduced probability density between the two peaks would not be representable at all with a single Gaussian.

The competitive distribution gets its name because its energy function is similar to the ones that govern winner-take-all networks[9]. When $N$ becomes large, the $N$ global minima of the energy function are singleton vectors (fig 3), with one component equal to unity, and the rest zero. This is due to a competitive interaction between the components. The mean of the zero temperature distribution is given by

$$\langle x_i \rangle = \frac{1}{N} \tag{8}$$

The eigenvalues of the covariance

$$\langle x_i x_j \rangle - \langle x_i \rangle \langle x_j \rangle = \frac{1}{N}\delta_{ij} - \frac{1}{N^2} \tag{9}$$

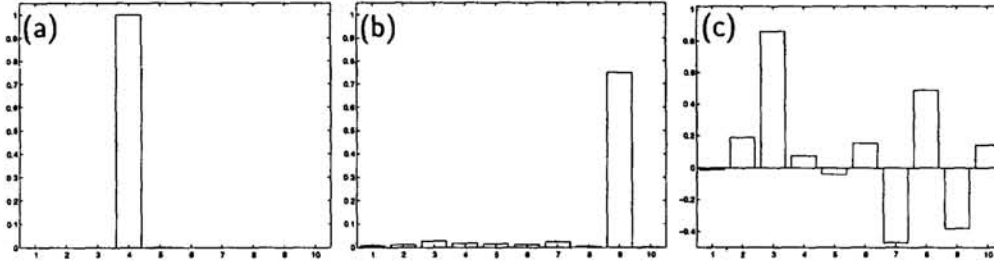

Figure 3: The competitive distribution for $N = 10$ variables. (a) One mode (zero temperature state) of the distribution. The strong competition between the variables results in only one variable on. There are $N$ modes of this form, each with a different winner variable. (b) A sample at finite temperature ($\beta \approx 110$) using Monte Carlo sampling. There is still a clear winner variable. (c) Sample from a standard Gaussian with matched mean and covariance. Even if we cut off the negative values this sample still bears little resemblance to the states shown in (a) and (b), since there is no clear winner variable.

all equal to $1/N$, except for a single zero mode. The zero mode is **1**, the vector of all ones, and the other eigenvectors span the $N - 1$ dimensional space perpendicular to **1**. Figure 3 shows two samples: one (b) drawn at finite temperature from the competitive distribution, and the other (c) drawn from a standard Gaussian distribution with the same mean and covariance. Even if the sample from the standard Gaussian is cut so negative values are set to zero the sample does not look at all like the original distribution. Most importantly a standard Gaussian will never be able to capture the strongly competitive character of this distribution.

## 5  COOPERATIVE DISTRIBUTION

To define the cooperative distribution on $N$ variables, an angle $\theta_i = 2\pi i/N$ is associated with each variable $x_i$, so that the variables can be regarded as sitting on a ring. The energy function is defined by

$$A_{ij} = \delta_{ij} + \frac{1}{N} - \frac{4}{N}\cos(\theta_i - \theta_j) \qquad (10)$$

$$b_i = 1; \qquad (11)$$

The coupling $A_{ij}$ between $x_i$ and $x_j$ depends only on the separation $\theta_i - \theta_j$ between them on the ring.

The minima, or ground states, of the energy function can be found numerically by the methods described earlier. An analytic calculation of the ground states in the large $N$ limit is also possible[5]. As shown in Figure 4(a), each ground state is a lump of activity centered at some angle on the ring. This delocalized pattern of activity is different from the singleton modes of the competitive distribution, and arises from the cooperative interactions between neurons on the ring. Because the distribution is invariant to rotations of the ring (cyclic permutations of the variables $x_i$), there are $N$ ground states, each with the lump at a different angle.

The mean and the covariance of the cooperative distribution are given by

$$\langle x_i \rangle = \text{const} \qquad (12)$$

$$\langle x_i x_j \rangle - \langle x_i \rangle \langle x_j \rangle = C(\theta_i - \theta_j) \qquad (13)$$

A given sample of $x$, shown in Figure 4(a), does not look anything like the mean, which is completely uniform. Samples generated from a Gaussian distribution with

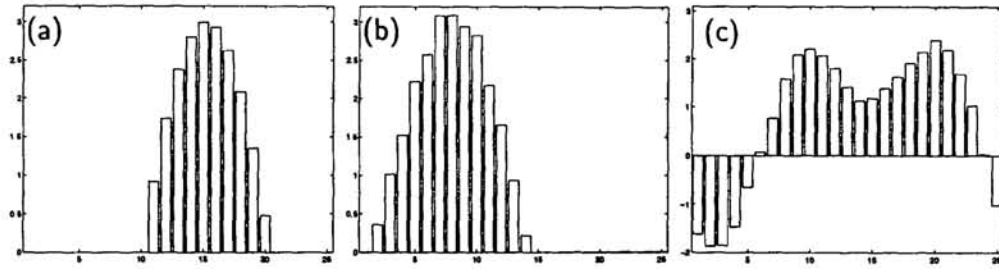

Figure 4: The cooperative distribution for $N = 25$ variables. (a) Zero temperature state. A cooperative interaction between the variables leads to a delocalized pattern of activity that can sit at different locations on the ring. (b) A finite temperature ($\beta = 50$) sample. (c) A sample from a standard Gaussian with matched mean and covariance.

the same mean and covariance look completely different from the ground states of the cooperative distribution (fig 4(c)).

These deviations from standard Gaussian behavior reflect fundamental differences in the underlying energy function. Here the energy function has $N$ discrete minima arranged along a ring. In the limit of large $N$ the barriers between these minima become quite small. A reasonable approximation is to regard the energy function as having a continuous line of minima with a ring geometry[5]. In other words, the energy surface looks like a curved trough, similar to the bottom of a wine bottle. The mean is the centroid of the ring and is not close to any minimum.

The cooperative distribution is able to model the set of all translations of the lump pattern of activity. This suggests that the rectified Gaussian may be useful in invariant object recognition, in cases where a continuous manifold of instantiations of an object must be modeled. One such case is visual object recognition, where the images of an object from different viewpoints form a continuous manifold.

## 6 SAMPLING

Figures 3 and 4 depict samples drawn from the competitive and cooperative distribution. These samples were generated using the Metropolis Monte Carlo algorithm. Since full descriptions of this algorithm can be found elsewhere, we give only a brief description of the particular features used here. The basic procedure is to generate a new configuration of the system and calculate the change in energy (given by eq. 2). If the energy decreases, one accepts the new configuration unconditionally. If it increases then the new configuration is accepted with probability $e^{-\beta \Delta E}$.

In our sampling algorithm one variable is updated at a time (analogous to single spin flips). The acceptance ratio is much higher this way than if we update all the spins simultaneously. However, for some distributions the energy function may have approximately marginal directions; directions in which there is little or no barrier. The cooperative distribution has this property. We can expect critical slowing down due to this and consequently some sort of collective update (analogous to multi-spin updates or cluster updates) might make sampling more efficient. However, the type of update will depend on the specifics of the energy function and is not easy to determine.

## 7   DISCUSSION

The competitive and cooperative distributions are examples of rectified Gaussians for which no good approximation by a standard Gaussian is possible. However, both distributions can be approximated by mixtures of standard Gaussians. The competitive distribution can be approximated by a mixture of $N$ Gaussians, one for each singleton state. The cooperative distribution can also be approximated by a mixture of $N$ Gaussians, one for each location of the lump on the ring. A more economical approximation would reduce the number of Gaussians in the mixture, but make each one anisotropic[10].

Whether the rectified Gaussian is superior to these mixture models is an empirical question that should be investigated empirically with specific real-world probabilistic modeling tasks. Our intuition is that the rectified Gaussian will turn out to be a good representation for nonlinear pattern manifolds, and the aim of this paper has been to make this intuition concrete.

To make the rectified Gaussian useful in practical applications, it is critical to find tractable learning algorithms. It is not yet clear whether learning will be more tractable for the rectified Gaussian than it was for the Boltzmann machine. Perhaps the continuous variables of the rectified Gaussian may be easier to work with than the binary variables of the Boltzmann machine.

**Acknowledgments**   We would like to thank P. Mitra, L. Saul, B. Shraiman and H. Sompolinsky for helpful discussions. Work on this project was supported by Bell Laboratories, Lucent Technologies.

## References

[1] D. H. Ackley, G. E. Hinton, and T. J. Sejnowski. A learning algorithm for Boltzmann machines. *Cognitive Science*, 9:147–169, 1985.

[2] G. E. Hinton and Z. Ghahramani. Generative models for discovering sparse distributed representations. *Phil. Trans. Roy. Soc.*, B352:1177–90, 1997.

[3] Z. Ghahramani and G. E. Hinton. Hierarchical non-linear factor analysis and topographic maps. *Adv. Neural Info. Proc. Syst.*, 11, 1998.

[4] H. S. Seung. How the brain keeps the eyes still. *Proc. Natl. Acad. Sci. USA*, 93:13339–13344, 1996.

[5] R. Ben-Yishai, R. L. Bar-Or, and H. Sompolinsky. Theory of orientation tuning in visual cortex. *Proc. Nat. Acad. Sci. USA*, 92:3844–3848, 1995.

[6] A. P. Georgopoulos, M. Taira, and A. Lukashin. Cognitive neurophysiology of the motor cortex. *Science*, 260:47–52, 1993.

[7] K. Zhang. Representation of spatial orientation by the intrinsic dynamics of the head-direction cell ensemble: a theory. *J. Neurosci.*, 16:2112–2126, 1996.

[8] D. P. Bertsekas. *Nonlinear programming*. Athena Scientific, Belmont, MA, 1995.

[9] S. Amari and M. A. Arbib. Competition and cooperation in neural nets. In J. Metzler, editor, *Systems Neuroscience*, pages 119–165. Academic Press, New York, 1977.

[10] G. E. Hinton, P. Dayan, and M. Revow. Modeling the manifolds of images of handwritten digits. *IEEE Trans. Neural Networks*, 8:65–74, 1997.